# Some new bounds on the generalization error of combined classifiers

**Vladimir Koltchinskii**
Department of Mathematics and Statistics
University of New Mexico
Albuquerque, NM 87131-1141
*vlad@math.unm.edu*

**Dmitriy Panchenko**
Department of Mathematics and Statistics
University of New Mexico
Albuquerque, NM 87131-1141
*panchenk@math.unm.edu*

**Fernando Lozano**
Department of Electrical and Computer Engineering
University of New Mexico
Albuquerque, NM 87131
*flozano@eece.unm.edu*

## Abstract

In this paper we develop the method of bounding the generalization error of a classifier in terms of its margin distribution which was introduced in the recent papers of Bartlett and Schapire, Freund, Bartlett and Lee. The theory of Gaussian and empirical processes allow us to prove the margin type inequalities for the most general functional classes, the complexity of the class being measured via the so called *Gaussian* complexity functions. As a simple application of our results, we obtain the bounds of Schapire, Freund, Bartlett and Lee for the generalization error of boosting. We also substantially improve the results of Bartlett on bounding the generalization error of neural networks in terms of $l_1-$norms of the weights of neurons. Furthermore, under additional assumptions on the complexity of the class of hypotheses we provide some tighter bounds, which in the case of boosting improve the results of Schapire, Freund, Bartlett and Lee.

## 1 Introduction and margin type inequalities for general functional classes

Let $(X, Y)$ be a random couple, where $X$ is an instance in a space $S$ and $Y \in \{-1, 1\}$ is a label. Let $\mathcal{G}$ be a set of functions from $S$ into $\mathbb{R}$. For $g \in \mathcal{G}$, $\mathrm{sign}(g(X))$ will be used as a predictor (a classifier) of the unknown label $Y$. If the distribution of $(X, Y)$ is unknown, then the choice of the predictor is based on the training data $(X_1, Y_1), \ldots, (X_n, Y_n)$ that consists of $n$ i.i.d. copies of $(X, Y)$. The goal of learning is to find a predictor $\hat{g} \in \mathcal{G}$ (based on the training data) whose *generalization (classification) error* $\mathbb{P}\{Y\hat{g}(X) \leq 0\}$ is small enough. We will first introduce some probabilistic bounds for general functional classes and then give several examples of their applications to bounding the generalization error of boosting and neural networks. We omit all the proofs and refer an interested reader to [5].

Let $(S, \mathcal{A}, P)$ be a probability space and let $\mathcal{F}$ be a class of measurable functions from $(S, \mathcal{A})$ into $\mathbb{R}$. Let $\{X_k\}$ be a sequence of i.i.d. random variables taking values in $(S, \mathcal{A})$ with common distribution $P$. Let $P_n$ be the empirical measure based on the sample $(X_1, \ldots, X_n)$, $P_n := n^{-1} \sum_{i=1}^{n} \delta_{X_i}$, where $\delta_x$ denotes the probability distribution concentrated at the point $x$. We will denote $Pf := \int_S f dP$, $P_n f := \int_S f dP_n$, etc. In what follows, $\ell^\infty(\mathcal{F})$ denotes the Banach space of uniformly bounded real valued functions on $\mathcal{F}$ with the norm $\|Y\|_{\mathcal{F}} := \sup_{f \in \mathcal{F}} |Y(f)|$, $Y \in \ell^\infty(\mathcal{F})$. Define

$$G_n(\mathcal{F}) := \mathbb{E} \|n^{-1} \sum_{i=1}^{n} g_i \delta_{X_i}\|_{\mathcal{F}} = \mathbb{E} \sup_{f \in \mathcal{F}} |n^{-1} \sum_{i=1}^{n} g_i f(X_i)|,$$

where $\{g_i\}$ is a sequence of i.i.d. standard normal random variables, independent of $\{X_i\}$. We will call $n \mapsto G_n(\mathcal{F})$ *the Gaussian complexity function* of the class $\mathcal{F}$. One can find in the literature (see, e.g. [11]) various upper bounds on such quantities as $G_n(\mathcal{F})$ in terms of entropies, VC-dimensions, etc.

We give below a bound in terms of *margin cost functions* (compare to [6, 7]) and Gaussian complexities.

Let $\Phi = \{\varphi_k : \mathbb{R} \to \mathbb{R}\}_{k=1}^{\infty}$ be a class of Lipschitz functions such that $(1 + \operatorname{sgn}(-x))/2 \leq \varphi_k(x)$ for all $x \in \mathbb{R}$ and all $k$. For each $\varphi \in \Phi$, $L(\varphi)$ will denote it's Lipschitz constant.

**Theorem 1** *For all $t > 0$,*

$$\mathbb{P}\{\exists f \in \mathcal{F} : P\{f \leq 0\} > \inf_{k \geq 1} \left[ P_n \varphi_k(f) + \sqrt{2\pi} L(\varphi_k) G_n(\mathcal{F}) \right.$$
$$\left. + \left( \frac{\log k}{n} \right)^{1/2} \right] + \frac{t+2}{\sqrt{n}} \} \leq 2 \exp\{-2t^2\}.$$

Let us consider a special family of cost functions. Assume that $\varphi$ is a fixed *nonincreasing* Lipschitz function from $\mathbb{R}$ into $\mathbb{R}$ such that $\varphi(x) \geq (1 + \operatorname{sgn}(-x))/2$ for all $x \in \mathbb{R}$. One can easily observe that $L(\varphi(\cdot/\delta)) \leq L(\varphi)\delta^{-1}$. Applying Theorem 1 to the class of Lipschitz functions $\Phi := \{\varphi(\cdot/\delta_k) : k \geq 0\}$, where $\delta_k := 2^{-k}$, we get the following result.

**Theorem 2** *For all $t > 0$,*

$$\mathbb{P}\{\exists f \in \mathcal{F} : P\{f \leq 0\} > \inf_{\delta \in [0,1]} \left[ P_n \varphi(\frac{f}{\delta}) + \frac{2\sqrt{2\pi} L(\varphi)}{\delta} G_n(\mathcal{F}) \right.$$
$$\left. + \left( \frac{\log \log_2(2\delta^{-1})}{n} \right)^{1/2} \right] + \frac{t+2}{\sqrt{n}} \} \leq 2 \exp\{-2t^2\}.$$

In [5] an example was given which shows that, in general, the order of the factor $\delta^{-1}$ in the second term of the bound can not be improved.

Given a metric space $(T, d)$, we denote $H_d(T; \varepsilon)$ the $\varepsilon$-entropy of $T$ with respect to $d$, i.e. $H_d(T; \varepsilon) := \log N_d(T; \varepsilon)$, where $N_d(T; \varepsilon)$ is the minimal number of balls of radius $\varepsilon$ covering $T$. The next theorem improves the previous results under some additional assumptions on the growth of random entropies $H_{d_{P_n,2}}(\mathcal{F}; \cdot)$. Define for $\gamma \in (0, 1]$

$$\delta_n(\gamma; f) := \sup\{\delta \in (0, 1) : \delta^\gamma P\{f \leq \delta\} \leq n^{-1 + \frac{\gamma}{2}}\}$$

and

$$\hat{\delta}_n(\gamma; f) := \sup\{\delta \in (0, 1) : \delta^\gamma P_n\{f \leq \delta\} \leq n^{-1 + \frac{\gamma}{2}}\}.$$

We call $\delta_n(\gamma; f)$ and $\hat{\delta}_n(\gamma; f)$, respectively, the *$\gamma$-margin* and the *empirical $\gamma$-margin* of $f$.

**Theorem 3** *Suppose that for some $\alpha \in (0,2)$ and for some constant $D > 0$*

$$H_{d_{P_n},2}(\mathcal{F}; u) \leq D u^{-\alpha}, \ u > 0 \text{ a.s.} \tag{1}$$

*Then for any $\gamma \geq \frac{2\alpha}{2+\alpha}$, for some constants $A, B > 0$ and for all large enough $n$*

$$\mathbb{P}\Big\{\forall f \in \mathcal{F} : A^{-1}\hat{\delta}_n(\gamma; f) \leq \delta_n(\gamma; f) \leq A\hat{\delta}_n(\gamma; f)\Big\}$$

$$\geq 1 - B(\log_2 \log_2 n) \exp\Big\{-n^{\frac{\gamma}{2}}/2\Big\}.$$

This implies that with high probability for all $f \in \mathcal{F}$

$$P\{f \leq 0\} \leq c(n^{1-\gamma/2}\hat{\delta}_n(\gamma; f)^\gamma)^{-1}.$$

The bound of Theorem 2 corresponds to the case of $\gamma = 1$. It is easy to see from the definitions of $\gamma$–margins that the quantity $(n^{1-\gamma/2}\hat{\delta}_n(\gamma; f)^\gamma)^{-1}$ increases in $\gamma \in (0,1]$. This shows that the bound in the case of $\gamma < 1$ is tighter. Further discussion of this type of bounds and their experimental study in the case of convex combinations of simple classifiers is given in the next section.

## 2  Bounding the generalization error of convex combinations of classifiers

Recently, several authors ([1, 8]) suggested a new class of upper bounds on generalization error that are expressed in terms of the empirical distribution of *the margin* of the predictor (the classifier). The margin is defined as the product $Y\hat{g}(X)$. The bounds in question are especially useful in the case of the classifiers that are the combinations of simpler classifiers (that belong, say, to a class $\mathcal{H}$). One of the examples of such classifiers is provided by the classifiers obtained by boosting [3, 4], bagging [2] and other voting methods of combining the classifiers. We will now demonstrate how our general results can be applied to the case of convex combinations of simple *base* classifiers.

We assume that $\tilde{S} := S \times \{-1, 1\}$ and $\tilde{\mathcal{F}} := \{\tilde{f} : f \in \mathcal{F}\}$, where $\tilde{f}(x, y) := yf(x)$. $P$ will denote the distribution of $(X, Y)$, $P_n$ the empirical distribution based on the observations $((X_1, Y_1), \ldots, (X_n, Y_n))$. It is easy to see that $G_n(\tilde{\mathcal{F}}) = G_n(\mathcal{F})$. One can easily see that if $\mathcal{F} := \text{conv}(\mathcal{H})$, where $\mathcal{H}$ is a class of base classifiers, then $G_n(\mathcal{F}) = G_n(\mathcal{H})$. These easy observations allow us to obtain useful bounds for boosting and other methods of combining the classifiers. For instance, we get in this case the following theorem that implies the bound of Schapire, Freund, Bartlett and Lee [8] when $\mathcal{H}$ is a VC-class of sets.

**Theorem 4** *Let $\mathcal{F} := \text{conv}(\mathcal{H})$, where $\mathcal{H}$ is a class of measurable functions from $(S, \mathcal{A})$ into $\mathbb{R}$. For all $t > 0$,*

$$\mathbb{P}\Big\{\exists f \in \mathcal{F} : P\{yf(x) \leq 0\} > \inf_{\delta \in [0,1]}\Big[P_n\varphi(\frac{yf(x)}{\delta}) + \frac{2\sqrt{2\pi}}{\delta}G_n(\mathcal{H})$$

$$+ \Big(\frac{\log\log_2(2\delta^{-1})}{n}\Big)^{1/2}\Big] + \frac{t+2}{\sqrt{n}}\Big\} \leq 2\exp\{-2t^2\}.$$

In particular, if $\mathcal{H}$ is a VC–class of classifiers $h : S \mapsto \{-1, 1\}$ (which means that the class of sets $\{\{x : h(x) = +1\} : h \in \mathcal{H}\}$ is a Vapnik–Chervonenkis class) with VC–dimension $V(\mathcal{H})$, we have with some constant $C > 0$, $G_n(\mathcal{H}) \leq C(V(\mathcal{H})/n)^{1/2}$. This implies that with probability at least $1 - \alpha$

$$P\{yf(x) \leq 0\} \leq \inf_{\delta \in (0,1]}\Big[P_n\{yf(x) \leq \delta\} + \frac{C}{\delta}\sqrt{\frac{V(\mathcal{H})}{n}} +$$

$$+\Big(\frac{\log\log_2(2\delta^{-1})}{n}\Big)^{1/2}\Big]+\frac{\sqrt{\frac12\log\frac2\alpha}+2}{\sqrt n},$$

which slightly improves the bound obtained previously by Schapire, Freund, Bartlett and Lee [8].

Theorem 3 provides some improvement of the above bounds on generalization error of convex combinations of base classifiers. To be specific, consider the case when $\mathcal{H}$ is a VC-class of classifiers. Let $V := V(\mathcal{H})$ be its VC-dimension. A well known bound (going back to Dudley) on the entropy of the convex hull (see [11], p. 142) implies that

$$H_{d_{P_n},2}(\mathrm{conv}(\mathcal{H}); u) \leq \sup_{Q\in\mathcal{P}(S)} H_{d_Q,2}(\mathrm{conv}(\mathcal{H}); u) \leq Du^{-\frac{2(V-1)}{V}}.$$

It immediately follows from Theorem 3 that for all $\gamma \geq \frac{2(V-1)}{2V-1}$ and for some constants $C, B$

$$\mathbb{P}\Big\{\exists f \in \mathrm{conv}(\mathcal{H}) : P\{\tilde f \leq 0\} > \frac{C}{n^{1-\gamma/2}\hat\delta_n(\gamma;f)^\gamma}\Big\} \leq B\log_2\log_2 n\exp\Big\{-\frac12 n^{\frac\gamma2}\Big\},$$

where

$$\hat\delta_n(\gamma;f) := \sup\Big\{\delta \in (0,1) : \delta^\gamma P_n\{(x,y) : yf(x) \leq \delta\} \leq n^{-1+\frac\gamma2}\Big\}.$$

This shows that in the case when the VC-dimension of the base is relatively small the generalization error of boosting and some other convex combinations of simple classifiers obtained by various versions of voting methods becomes better than it was suggested by the bounds of Schapire, Freund, Bartlett and Lee. One can also conjecture that the remarkable generalization ability of these methods observed in numerous experiments can be related to the fact that the combined classifier belongs to *a subset* of the convex hull for which the random entropy $H_{d_{P_n},2}$ is much smaller than for the whole convex hull (see [9, 10] for improved margin type bounds in a much more special setting).

To demonstrate the improvement provided by our bounds over previous results, we show some experimental evidence obtained for a simple artificially generated problem, for which we are able to compute exactly the generalization error as well as the $\gamma$-margins.

We consider the problem of learning a classifier consisting of the indicator function of the union of a finite number of intervals in the input space $S = [0,1]$. We used the Adaboost algorithm [4] to find a combined classifier using as base class $\mathcal{H} = \{[0,b] : b \in [0,1]\} \cup \{[b,1] : b \in [0,1]\}$ (i.e. decision stumps). Notice that in this case $V = 2$, and according to the theory values of gamma in $(2/3,1)$ should result in tighter bounds on the generalization error.

For our experiments we used a target function with 10 equally spaced intervals, and a sample size of 1000, generated according to the uniform distribution in $[0,1]$. We ran Adaboost for 500 rounds, and computed at each round the generalization error of the combined classifier and the bound $C(n^{1-\gamma/2}\hat\delta_n(\gamma;f)^\gamma)^{-1}$ for different values of $\gamma$. We set the constant $C$ to one.

In figure 1 we plot the generalization error and the bounds for $\gamma = 1, 0.8$ and $2/3$. As expected, for $\gamma = 1$ (which corresponds roughly to the bounds in [8]) the bound is very loose, and as $\gamma$ decreases, the bound gets closer to the generalization error. In figure 2 we show that by reducing further the value of $\gamma$ we get a curve even closer to the actual generalization error (although for $\gamma = 0.2$ we do not get an upper bound). This seems to support the conjecture that Adaboost generates combined classifiers that belong to a subset of of the convex hull of $\mathcal{H}$ with a smaller random entropy. In figure 3 we plot the ratio $\hat\delta_n(\gamma;f)/\delta_n(\gamma;f)$ for $\gamma = 0.4, 2/3$ and $0.8$ against the boosting iteration. We can see that the ratio is close to one in all the examples indicating that the value of the constant $A$ in theorem 3 is close to one in this case.

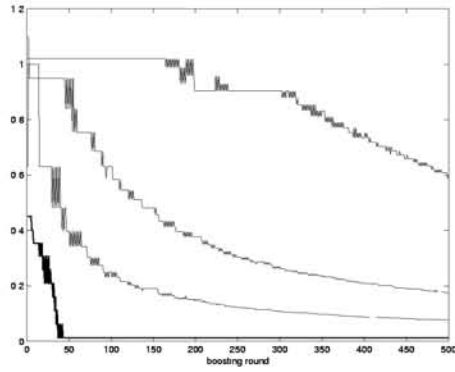

Figure 1: Comparison of the generalization error (thicker line) with $(n^{1-\gamma/2}\hat{\delta}_n(\gamma;f)^\gamma)^{-1}$ for $\gamma = 1, 0.8$ and $2/3$ (thinner lines, top to bottom).

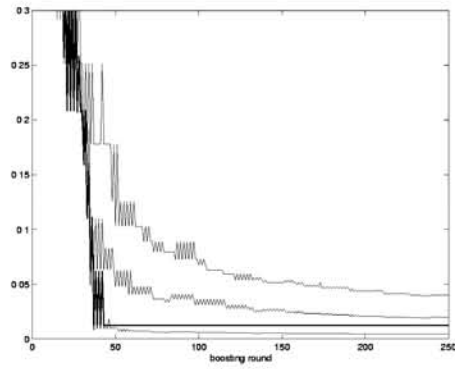

Figure 2: Comparison of the generalization error (thicker line) with $(n^{1-\gamma/2}\hat{\delta}_n(\gamma;f)^\gamma)^{-1}$ for $\gamma = 0.5, 0.4$ and $0.2$ (thinner lines, top to bottom).

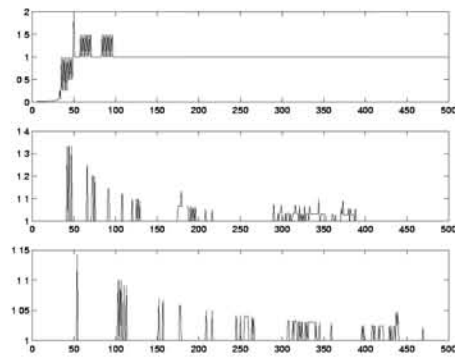

Figure 3: Ratio $\hat{\delta}_n(\gamma;f)/\delta_n(\gamma;f)$ versus boosting round for $\gamma = 0.4, 2/3, 0.8$ (top to bottom)

## 3   Bounding the generalization error in neural network learning

We turn now to the applications of the bounds of previous section in neural network learning. Let $\mathcal{H}$ be a class of measurable functions from $(S, \mathcal{A})$ into $\mathbb{R}$. Given a sigmoid $\sigma$ from $\mathbb{R}$ into $[-1, 1]$ and a vector $w := (w_1, \ldots, w_n) \in \mathbb{R}^n$, let $N_{\sigma, w}(u_1, \ldots, u_n) := \sigma(\sum_{i=1}^n w_j u_j)$. We call the function $N_{\sigma, w}$ *a neuron* with weights $w$ and sigmoid $\sigma$. For $w \in \mathbb{R}^n$, $\|w\|_{\ell_1} := \sum_{i=1}^n |w_i|$. Let $\sigma_j : j \geq 1$ be functions from $\mathbb{R}$ into $[-1, 1]$, satisfying the Lipschitz conditions:

$$|\sigma_j(u) - \sigma_j(v)| \leq L_j |u - v|, \; u, v \in \mathbb{R}.$$

Let $\{A_j\}$ be a sequence of positive numbers. We define recursively classes of neural networks with restrictions on the weights of neurons ($j$ below is the number of layers):

$$\mathcal{H}_0 = \mathcal{H}, \; \mathcal{H}_j(A_1, \ldots, A_j) :=$$

$$:= \Big\{ N_{\sigma_j, w}(h_1, \ldots, h_n) : n \geq 0, h_i \in \mathcal{H}_{j-1}(A_1, \ldots, A_{j-1}), \; w \in \mathbb{R}^n, \|w\|_{\ell_1} \leq A_j \Big\} \bigcup$$

$$\bigcup \mathcal{H}_{j-1}(A_1, \ldots, A_{j-1}).$$

**Theorem 5**   *For all $t > 0$ and for all $l \geq 1$*

$$\mathbb{P}\Big\{ \exists f \in \mathcal{H}_l(A_1, \ldots, A_l) : P\{\tilde{f} \leq 0\} > \inf_{\delta \in (0,1]} \Big[ P_n \varphi(\frac{\tilde{f}}{\delta}) + \frac{1}{\delta} \prod_{k=1}^l (2L_j A_j + 1) G_n(\mathcal{H}) +$$

$$+ \Big( \frac{\log \log_2(2\delta^{-1})}{n} \Big)^{1/2} + \frac{t+2}{\sqrt{n}} \Big\} \leq 2 \exp\{-2t^2\}$$

**Remark.** Bartlett [1] obtained a similar bound for a more special class $\mathcal{H}$ and with larger constants. In the case when $A_j \equiv A, L_j \equiv L$ (the case considered by Bartlett) the expression in the right hand side of his bound includes $\frac{(AL)^{l(l+1)/2}}{\delta^l}$, which is replaced in our bound by $\frac{(AL)^l}{\delta}$. These improvement can be substantial in applications, since the above quantities play the role of complexity penalties.

Finally, it is worth mentioning that the theorems of Section 1 can be applied also to bounding the generalization error in multi-class problems. Namely, we assume that the labels take values in a finite set $\mathcal{Y}$ with $\mathrm{card}(\mathcal{Y}) =: L$. Consider a class $\tilde{\mathcal{F}}$ of functions from $\tilde{S} := S \times \mathcal{Y}$ into $\mathbb{R}$. A function $f \in \tilde{\mathcal{F}}$ predicts a label $y \in \mathcal{Y}$ for an example $x \in S$ iff

$$f(x, y) > \max_{y' \neq y} f(x, y').$$

The margin of an example $(x, y)$ is defined as

$$m_f(x, y) := f(x, y) - \max_{y' \neq y} f(x, y'),$$

so $f$ misclassifies the example $(x, y)$ iff $m_f(x, y) \leq 0$. Let

$$\mathcal{F} := \{f(\cdot, y) : y \in \mathcal{Y}, f \in \tilde{\mathcal{F}}\}.$$

The next result follows from Theorem 2.

**Theorem 6**   *For all $t > 0$,*

$$\mathbb{P}\Big\{ \exists f \in \tilde{\mathcal{F}} : P\{m_f \leq 0\} > \inf_{\delta \in (0,1]} \Big[ P_n\{m_f \leq \delta\} + \frac{4\sqrt{2\pi} L(2L-1)}{\delta} G_n(\mathcal{F}) +$$

$$+ \Big( \frac{\log \log_2(2\delta^{-1})}{n} \Big)^{1/2} + \frac{t+2}{\sqrt{n}} \Big\} \leq 2 \exp\{-2t^2\}.$$

# References

[1] Bartlett, P. (1998) The Sample Complexity of Pattern Classification with Neural Networks: The Size of the Weights is More Important than the Size of the Network. *IEEE Transactions on Information Theory*, 44, 525-536.

[2] Breiman, L. (1996). Bagging Predictors. *Machine Learning,*26(2), 123-140.

[3] Freund Y. (1995) Boosting a weak learning algorithm by majority. *Information and Computation,*121,2,256-285.

[4] Freund Y. and Schapire, R.E. (1997) A decision-theoretic generalization of on-line learning and an application to boosting. *Journal of Computer and System Sciences*, 55(1),119-139.

[5] Koltchinskii, V. and Panchenko, D. (2000) Empirical margin distributions and bounding the generalization error of combined classifiers, preprint.

[6] Mason, L., Bartlett, P. and Baxter, J. (1999) Improved Generalization through Explicit Optimization of Margins. *Machine Learning* , 0, 1-11.

[7] Mason, L., Baxter, J., Bartlett, P. and Frean, M. (1999) Functional Gradient Techniques for Combining Hypotheses. In: Advances in Large Margin Classifiers. Smola, Bartlett, Schölkopf and Schnurmans (Eds), to appear.

[8] Schapire, R., Freund, Y., Bartlett, P. and Lee, W. S. (1998) Boosting the Margin: A New Explanation of Effectiveness of Voting Methods. *Ann. Statist.*, 26, 1651-1687.

[9] Shawe–Taylor, J. and Cristianini, N. (1999) Margin Distribution Bounds on Generalization. In: Lecture Notes in Artificial Intelligence, 1572. Computational Learning Theory, 4th European Conference, EuroCOLT'99, 263–273.

[10] Shawe–Taylor, J. and Cristianini, N. (1999) Further Results on the Margin Distribution. Proc. of COLT'99, 278–285.

[11] van der Vaart, A. and Wellner, J. (1996) Weak convergence and Empirical Processes. With Applications to Statistics. Springer-Verlag, New York.
